# Object Detection with Grammar Models

**Ross B. Girshick**
Dept. of Computer Science
University of Chicago
Chicago, IL 60637
rbg@cs.uchicago.edu

**Pedro F. Felzenszwalb**
School of Engineering and
Dept. of Computer Science
Brown University
Providence, RI 02912
pff@brown.edu

**David McAllester**
TTI-Chicago
Chicago, IL 60637
mcallester@ttic.edu

## Abstract

Compositional models provide an elegant formalism for representing the visual appearance of highly variable objects. While such models are appealing from a theoretical point of view, it has been difficult to demonstrate that they lead to performance advantages on challenging datasets. Here we develop a grammar model for person detection and show that it outperforms previous high-performance systems on the PASCAL benchmark. Our model represents people using a hierarchy of deformable parts, variable structure and an explicit model of occlusion for partially visible objects. To train the model, we introduce a new discriminative framework for learning structured prediction models from weakly-labeled data.

## 1 Introduction

The idea that images can be hierarchically parsed into objects and their parts has a long history in computer vision, see for example [15]. Image parsing has also been of considerable recent interest [11, 13, 21, 22, 24]. However, it has been difficult to demonstrate that sophisticated compositional models lead to performance advantages on challenging metrics such as the PASCAL object detection benchmark [9]. In this paper we achieve new levels of performance for person detection using a grammar model that is richer than previous models used in high-performance systems. We also introduce a general framework for learning discriminative models from weakly-labeled data.

Our models are based on the object detection grammar formalism in [11]. Objects are represented in terms of other objects through compositional rules. Deformation rules allow for the parts of an object to move relative to each other, leading to hierarchical deformable part models. Structural variability provides choice between multiple part subtypes — effectively creating mixture models throughout the compositional hierarchy — and also enables optional parts. In this formalism parts may be reused both within an object category and across object categories.

Our baseline and departure point is the UoC-TTI object detector [10, 12]. This system represents a class of objects with three different pictorial structure models. Although these models are learned automatically, making semantic interpretation unclear, it seems that the three components for the person class differ in how much of the person is taken to be visible — just the head and shoulders, the head and shoulders together with the upper body, or the whole standing person. Each of the three components has independently trained parts. For example, each component has a head part trained independently from the head part of the other components.

Here we construct a single grammar model that allows more flexibility in describing the amount of the person that is visible. The grammar model avoids dividing the training data between different components and thus uses the training data more efficiently. The parts in the model, such as the head part, are shared across different interpretations of the degree of visibility of the person. The grammar model also includes subtype choice at the part level to accommodate greater appearance

variability across object instances. We use parts with subparts to benefit from high-resolution image data, while also allowing for deformations. Unlike previous approaches, we explicitly model the source of occlusion for partially visible objects.

Our approach differs from that of Jin and Geman [13] in that theirs focuses on whole scene interpretation with generative models, while we focus on discriminatively trained models of individual objects. We also make Markovian restrictions not made in [13]. Our work is more similar to that of Zhu et al. [21] who impose similar Markovian restrictions. However, our training method, image features, and grammar design are substantially different.

The model presented here is designed to accurately capture the visible portion of a person. There has been recent related work on occlusion modeling in pedestrian and person images [7, 18]. In [7], Enzweiler et al. assume access to depth and motion information in order to estimate occlusion boundaries. In [18], Wang et al. rely on the observation that the scores of individual filter cells (using the Dalal and Triggs detector [5]) can reliably predict occlusion in the INRIA pedestrian data. This does not hold for the harder PASCAL person data.

In addition to developing a grammar model for detecting people, we develop new training methods which contribute to our boost in performance. Training data for vision is often assigned weak labels such as bounding boxes or just the names of objects occurring in the image. In contrast, an image analysis system will often produce strong predictions such as a segmentation or a pose. Existing structured prediction methods, such as structural SVM [16, 17] and latent structural SVM [19], do not directly support weak labels together with strong predictions. We develop the notion of a *weak-label structural SVM* which generalizes structural SVMs and latent-structural SVMs. The key idea is to introduce a loss $L(y, s)$ for making a strong prediction $s$ when the weak training label is $y$.

A formalism for learning from weak labels was also developed in [2]. One important difference is that [2] generalizes ranking SVMs.[1] Our framework also allows for softer relations between weak labels and strong predictions.

## 2  Grammar models

Object detection grammars [11] represent objects recursively in terms of other objects. Let $\mathcal{N}$ be a set of nonterminal symbols and $\mathcal{T}$ be a set of terminal symbols. We can think of the terminals as the basic building blocks that can be found in an image. The nonterminals define abstract objects whose appearance are defined in terms of expansions into terminals.

Let $\Omega$ be a set of possible locations for a symbol within an image. A placed symbol, $Y(\omega)$, specifies a placement of $Y \in \mathcal{N} \cup \mathcal{T}$ at a location $\omega \in \Omega$. The structure of a grammar model is defined by a set, $R$, of weighted productions of the form

$$X(\omega_0) \xrightarrow{s} \{ Y_1(\omega_1), \ldots, Y_n(\omega_n) \}, \tag{1}$$

where $X \in \mathcal{N}$, $Y_i \in \mathcal{N} \cup \mathcal{T}$, $\omega_i \in \Omega$ and $s \in \mathbb{R}$ is a score. We denote the score of $r \in R$ by $s(r)$.

We can expand a placed nonterminal to a bag of placed terminals by repeatedly applying productions. An expansion of $X(\omega)$ leads to a derivation tree $T$ rooted at $X(\omega)$. The leaves of $T$ are labeled with placed terminals, and the internal nodes of $T$ are labeled with placed nonterminals and with the productions used to replace those symbols.

We define appearance models for the terminals using a function $\mathrm{score}(A, \omega)$ that computes a score for placing the terminal $A$ at location $\omega$. This score depends implicitly on the image data. We define the score of a derivation tree $T$ to be the sum of the scores of the productions used to generate $T$, plus the score of placing the terminals associated with $T$'s leaves in their respective locations.

$$\mathrm{score}(T) = \sum_{r \in \mathrm{internal}(T)} s(r) + \sum_{A(w) \in \mathrm{leaves}(T)} \mathrm{score}(A, \omega) \tag{2}$$

To generalize the models from [10] we let $\Omega$ be positions and scales within a feature map pyramid $H$. We define the appearance models for terminals by associating a filter $F_A$ with each terminal $A$.

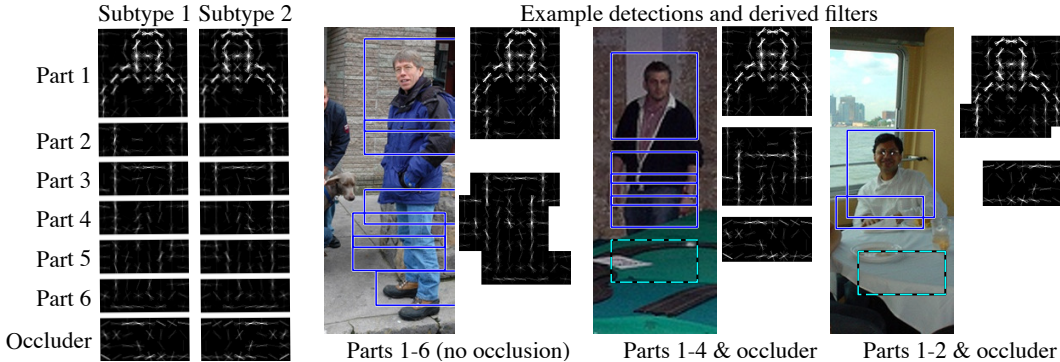

| | Subtype 1 | Subtype 2 | Example detections and derived filters |
|---|---|---|---|

Figure 1: *Shallow grammar model.* This figure illustrates a shallow version of our grammar model (Section 2.1). This model has six person parts and an occlusion model ("occluder"), each of which comes in one of two subtypes. A detection places one subtype of each visible part at a location and scale in the image. If the derivation does not place all parts it must place the occluder. Parts are allowed to move relative to each other, but their displacements are constrained by deformation penalties.

Then $\text{score}(A,\omega) = F_A \cdot \phi(H,\omega)$ is the dot product between the filter coefficients and the features in a subwindow of the feature map pyramid, $\phi(H,\omega)$. We use the variant of histogram of oriented gradient (HOG [5]) features described in [10].

We consider models with productions specified by two kinds of schemas (a schema is a template for generating productions). A structure schema specifies one production for each placement $\omega \in \Omega$,

$$X(\omega) \xrightarrow{s} \{\, Y_1(\omega \oplus \delta_1), \ldots, Y_n(\omega \oplus \delta_n) \,\}. \tag{3}$$

Here the $\delta_i$ specify constant displacements within the feature map pyramid. Structure schemas can be used to define decompositions of objects into other objects.

Let $\Delta$ be the set of possible displacements within a single scale of a feature map pyramid. A deformation schema specifies one production for each placement $\omega \in \Omega$ and displacement $\delta \in \Delta$,

$$X(\omega) \xrightarrow{\alpha \cdot \phi(\delta)} \{\, Y(\omega \oplus \delta) \,\}. \tag{4}$$

Here $\phi(\delta)$ is a feature vector and $\alpha$ is a vector of deformation parameters. Deformation schemas can be used to define deformable models. We define $\phi(\delta) = (dx, dy, dx^2, dy^2)$ so that deformation scores are quadratic functions of the displacements.

The parameters of our models are defined by a weight vector $\mathbf{w}$ with entries for the score of each structure schema, the deformation parameters of each deformation schema and the filter coefficients associated with each terminal. Then $\text{score}(T) = \mathbf{w} \cdot \Phi(T)$, where $\Phi(T)$ is the sum of (sparse) feature vectors associated with each placed terminal and production in $T$.

## 2.1 A grammar model for detecting people

Each component in the person model learned by the `voc-release4` system [12] is tuned to detect people under a prototypical visibility pattern. Based on this observation we designed, by hand, the structure of a grammar that models visibility by using structural variability and optional parts. For clarity, we begin by describing a shallow model (Figure 1) that places all filters at the same resolution in the feature map pyramid. After explaining this model, we describe a deeper model that includes deformable subparts at higher resolutions.

**Fine-grained occlusion** Our grammar model has a start symbol $Q$ that can be expanded using one of six possible structure schemas. These choices model different degrees of visibility ranging from heavy occlusion (only the head and shoulders are visible) to no occlusion at all.

Beyond modeling fine-grained occlusion patterns when compared to the mixture models from [7] and [12], our grammar model is also richer in a number of ways. In Section 5 we show that each of the following modeling choices improves detection performance.

**Occlusion model** If a person is occluded, then there must be some cause of the occlusion — either the edge of the image or an occluding object, such as a desk or dinner table. We use a nontrivial model to capture the appearance of the *stuff* that occludes people.

**Part subtypes** The mixture model from [12] has two subtypes for each mixture component. The subtypes are forced to be mirror images of each other and correspond roughly to left-facing versus right-facing people. Our grammar model has two subtypes for each part, which are also forced to be mirror images of each other. But in the case of our grammar model, the decision of which part subtype to instantiate at detection time is independent for each part.

The shallow person grammar model is defined by the following grammar. The indices $p$ (for part), $t$ (for subtype), and $k$ have the following ranges: $p \in \{1, \ldots, 6\}$, $t \in \{L, R\}$ and $k \in \{1, \ldots, 5\}$.

$$
\begin{aligned}
Q(\omega) &\xrightarrow{s_k} \{ Y_1(\omega \oplus \delta_1), \ldots, Y_k(\omega \oplus \delta_k), O(\omega \oplus \delta_{k+1}) \} \\
Q(\omega) &\xrightarrow{s_6} \{ Y_1(\omega \oplus \delta_1), \ldots, Y_6(\omega \oplus \delta_6) \} \\
Y_p(\omega) \xrightarrow{0} \{ Y_{p,t}(\omega) \} \quad &Y_{p,t}(\omega) \xrightarrow{\alpha_{p,t} \cdot \phi(\delta)} \{ A_{p,t}(\omega \oplus \delta) \} \\
O(\omega) \xrightarrow{0} \{ O_t(\omega) \} \quad &O_t(\omega) \xrightarrow{\alpha_t \cdot \phi(\delta)} \{ A_t(\omega \oplus \delta) \}
\end{aligned}
$$

The grammar has a start symbol $Q$ with six alternate choices that derive people under varying degrees of visibility (occlusion). Each part has a corresponding nonterminal $Y_p$ that is placed at some ideal position relative to $Q$. Derivations with occlusion include the occlusion symbol $O$. A derivation selects a subtype and displacement for each visible part. The parameters of the grammar (production scores, deformation parameters and filters) are learned with the discriminative procedure described in Section 4. Figure 1 illustrates the filters in the resulting model and some example detections.

**Deeper model** We extend the shallow model by adding deformable subparts at two scales: (1) the same as, and (2) twice the resolution of the start symbol $Q$. When detecting large objects, high-resolution subparts capture fine image details. However, when detecting small objects, high-resolution subparts cannot be used because they "fall off the bottom" of the feature map pyramid. The model uses derivations with low-resolution subparts when detecting small objects.

We begin by *replacing* the productions from $Y_{p,t}$ in the grammar above, and then adding new productions. Recall that $p$ indexes the top-level parts and $t$ indexes subtypes. In the following schemas, the indices $r$ (for resolution) and $u$ (for subpart) have the ranges: $r \in \{H, L\}$, $u \in \{1, \ldots, N_p\}$, where $N_p$ is the number of subparts in a top-level part $Y_p$.

$$
\begin{aligned}
Y_{p,t}(\omega) &\xrightarrow{\alpha_{p,t} \cdot \phi(\delta)} \{ Z_{p,t}(\omega \oplus \delta) \} \\
Z_{p,t}(\omega) &\xrightarrow{0} \{ A_{p,t}(\omega), W_{p,t,r,1}(\omega \oplus \delta_{p,t,r,1}), \ldots, W_{p,t,r,N_p}(\omega \oplus \delta_{p,t,r,N_p}) \} \\
W_{p,t,r,u}(\omega) &\xrightarrow{\alpha_{p,t,r,u} \cdot \phi(\delta)} \{ A_{p,t,r,u}(\omega \oplus \delta) \}
\end{aligned}
$$

We note that as in [23] our model has hierarchical deformations. The part terminal $A_{p,t}$ can move relative to $Q$ and the subpart terminal $A_{p,t,r,u}$ can move relative to $A_{p,t}$.

The displacements $\delta_{p,t,H,u}$ place the symbols $W_{p,t,H,u}$ one octave below $Z_{p,t}$ in the feature map pyramid. The displacements $\delta_{p,t,L,u}$ place the symbols $W_{p,t,L,u}$ at the same scale as $Z_{p,t}$. We add subparts to the first two top-level parts ($p = 1$ and 2), with the number of subparts set to $N_1 = 3$ and $N_2 = 2$. We find that adding additional subparts does not improve detection performance.

## 2.2 Inference and test time detection

Inference involves finding high scoring derivations. At test time, because images may contain multiple instances of an object class, we compute the maximum scoring derivation rooted at $Q(\omega)$, for each $\omega \in \Omega$. This can be done efficiently using a standard dynamic programming algorithm [11].

We retain only those derivations that score above a threshold, which we set low enough to ensure high recall. We use $\text{box}(T)$ to denote a detection window associated with a derivation $T$. Given a set of candidate detections, we apply nonmaximal suppression to produce a final set of detections.

We define $\text{box}(T)$ by assigning a detection window size, in feature map coordinates, to each structure schema that can be applied to $Q$. This leads to detections with one of six possible aspect ratios, depending on which production was used in the first step of the derivation. The absolute location and size of a detection depends on the placement of $Q$. For the first five production schemas, the ideal location of the occlusion part, $O$, is *outside* of $\text{box}(T)$.

## 3 Learning from weakly-labeled data

Here we define a general formalism for learning functions from weakly-labeled data. Let $\mathcal{X}$ be an input space, $\mathcal{Y}$ be a label space, and $\mathcal{S}$ be an output space. We are interested in learning functions $f : \mathcal{X} \rightarrow \mathcal{S}$ based on a set of training examples $\{(x_1, y_1), \ldots, (x_n, y_n)\}$ where $(x_i, y_i) \in \mathcal{X} \times \mathcal{Y}$. In contrast to the usual supervised learning setting, we do not assume that the label space and the output space are the same. In particular there may be many output values that are compatible with a label, and we can think of each example as being only weakly labeled. It will also be useful to associate a subset of possible outputs, $\mathcal{S}(x) \subseteq \mathcal{S}$, with an example $x$. In this case $f(x) \in \mathcal{S}(x)$.

A connection between labels and outputs can be made using a loss function $L : \mathcal{Y} \times \mathcal{S} \rightarrow \mathbb{R}$. $L(y, s)$ associates a cost with the prediction $s \in \mathcal{S}$ on an example labeled $y \in \mathcal{Y}$. Let $\mathcal{D}$ be a distribution over $\mathcal{X} \times \mathcal{Y}$. Then a natural goal is to find a function $f$ with low expected loss $E_{\mathcal{D}}[L(y, f(x))]$.

A simple example of a weakly-labeled training problem comes from learning sliding window classifiers in the PASCAL object detection dataset. The training data specifies pixel-accurate bounding boxes for the target objects while a sliding window classifier reports boxes with a fixed aspect ratio and at a finite number of scales. The output space is, therefore, a subset of the label space.

As usual, we assume $f$ is parameterized by a vector of model parameters $\mathbf{w}$ and generates predictions by maximizing a linear function of a joint feature map $\Phi(x, s)$, $f(x) = \operatorname{argmax}_{s \in \mathcal{S}(x)} \mathbf{w} \cdot \Phi(x, s)$.

We can train $\mathbf{w}$ by minimizing a regularized risk on the training set. We define a *weak-label structural SVM* (WL-SSVM) by the following training equation,

$$E(\mathbf{w}) = \frac{1}{2}||\mathbf{w}||^2 + C \sum_{i=1}^{n} L'(\mathbf{w}, x_i, y_i). \tag{5}$$

The surrogate training loss $L'$ is defined in terms of two different loss augmented predictions.

$$L'(\mathbf{w}, x, y) = \underbrace{\max_{s \in \mathcal{S}(x)} \left[ \mathbf{w} \cdot \Phi(x, s) + L_{\mathrm{margin}}(y, s) \right]}_{(6a)} - \underbrace{\max_{s \in \mathcal{S}(x)} \left[ \mathbf{w} \cdot \Phi(x, s) - L_{\mathrm{output}}(y, s) \right]}_{(6b)} \tag{6}$$

$L_{\mathrm{margin}}$ encourages high-loss outputs to "pop out" of (6a), so that their scores get pushed down. $L_{\mathrm{output}}$ suppresses high-loss outputs in (6b), so the score of a low-loss prediction gets pulled up.

It is natural to take $L_{\mathrm{margin}} = L_{\mathrm{output}} = L$. In this case $L'$ becomes a type of ramp loss [4, 6, 14]. Alternatively, taking $L_{\mathrm{margin}} = L$ and $L_{\mathrm{output}} = 0$ gives the ramp loss that has been shown to be consistent in [14]. As we discuss below, the choice of $L_{\mathrm{output}}$ can have a significant effect on the computational difficulty of the training problem. Several popular learning frameworks can be derived as special cases of WL-SSVM. For the examples below, let $\mathbf{I}(a, b) = 0$ when $a = b$, and $\mathbf{I}(a, b) = \infty$ when $a \neq b$.

**Structural SVM** Let $\mathcal{S} = \mathcal{Y}$, $L_{\mathrm{margin}} = L$ and $L_{\mathrm{output}}(y, \hat{y}) = \mathbf{I}(y, \hat{y})$. Then $L'(\mathbf{w}, x, y)$ is the hinge loss used in a structural SVM [17]. In this case $L'$ is convex in $\mathbf{w}$ because the maximization in (6b) disappears. We note, however, that this choice of $L_{\mathrm{output}}$ may be problematic and lead to inconsistent training problems. Consider the following situation. A training example $(x, y)$ may be compatible with a different label $\hat{y} \neq y$, in the sense that $L(y, \hat{y}) = 0$. But even in this case a structural SVM pushes the score $\mathbf{w} \cdot \Phi(x, y)$ to be above $\mathbf{w} \cdot \Phi(x, \hat{y})$. This issue can be addressed by relaxing $L_{\mathrm{output}}$ to include a maximization over labels in (6b).

**Latent structural SVM** Now let $\mathcal{Z}$ be a space of latent values, $\mathcal{S} = \mathcal{Y} \times \mathcal{Z}$, $L_{\mathrm{margin}} = L$ and $L_{\mathrm{output}}(y, (\hat{y}, \hat{z})) = \mathbf{I}(y, \hat{y})$. Then $L'(\mathbf{w}, x, y)$ is the hinge loss used in a latent structural SVM [19]. In this case $L'$ is not convex in $\mathbf{w}$ due to the maximization over latent values in (6b). As in the previous example, this choice of $L_{\mathrm{output}}$ can be problematic because it "requires" that the training labels be predicted exactly. This can be addressed by relaxing $L_{\mathrm{output}}$, as in the previous example.

## 4 Training grammar models

Now we consider learning the parameters of an object detection grammar using the training data in the PASCAL VOC datasets with the WL-SSVM framework. For two rectangles $a$ and $b$ let $\operatorname{overlap}(a, b) = \operatorname{area}(a \cap b) / \operatorname{area}(a \cup b)$. We will use this measure of overlap in our loss functions.

For training, we augment our model's output space (the set of all derivation trees), with the background output $\perp$. We define $\Phi(x, \perp)$ to be the zero vector, as was done in [1]. Thus the score of a background hypothesis is zero independent of the model parameters $\mathbf{w}$.

The training data specifies a bounding box for each instance of an object in a set of training images. We construct a set of weakly-labeled examples $\{(x_1, y_1), \ldots, (x_n, y_n)\}$ as follows. For each training image $I$, and for each bounding box $B$ in $I$, we define a *foreground* example $(x, y)$, where $y = B$, $x$ specifies the image $I$, and the set of valid predictions $\mathcal{S}(x)$ includes:

1. Derivations $T$ with $\text{overlap}(\text{box}(T), B) \geq 0.1$ *and* $\text{overlap}(\text{box}(T), B') < 0.5$ for all $B'$ in $I$ such that $B' \neq B$.
2. The background output $\perp$.

The overlap requirements in (1) ensure that we consider only predictions that are relevant for a particular object instance, while avoiding interactions with other objects in the image.

We also define a very large set of *background* examples. For simplicity, we use images that do not contain any bounding boxes. For each background image $I$, we define a different example $(x, y)$ for each position and scale $\omega$ within $I$. In this case $y = \perp$, $x$ specifies the image $I$, and $\mathcal{S}(x)$ includes derivations $T$ rooted at $Q(\omega)$ and the background output $\perp$. The set of background examples is very large because the number of positions and scales within each image is typically around 250K.

## 4.1 Loss functions

The PASCAL benchmark requires a correct detection to have at least 50% overlap with a ground-truth bounding box. We use this rule to define our loss functions. First, define $L_{l,\tau}(y, s)$ as follows

$$L_{l,\tau}(y, s) = \begin{cases} l & \text{if } y = \perp \text{ and } s \neq \perp \\ 0 & \text{if } y = \perp \text{ and } s = \perp \\ l & \text{if } y \neq \perp \text{ and } \text{overlap}(y, s) < \tau \\ 0 & \text{if } y \neq \perp \text{ and } \text{overlap}(y, s) \geq \tau. \end{cases} \tag{7}$$

Following the PASCAL VOC protocol we use $L_{\text{margin}} = L_{1,0.5}$. For a foreground example this pushes down the score of detections that don't overlap with the bounding box label by at least 50%.

Instead of using $L_{\text{output}} = L_{\text{margin}}$, we let $L_{\text{output}} = L_{\infty,0.7}$. For a foreground example this ensures that the maximizer of (6b) is a detection with high overlap with the bounding box label. For a background example, the maximizer of (6b) is always $\perp$. Later we discuss how this simplifies our optimization algorithm. While our choice of $L_{\text{output}}$ does not produce a convex objective, it does tightly limit the range of outputs, making our optimization less prone to reaching bad local optima.

## 4.2 Optimization

Since $L'$ is not convex, the WL-SSVM objective (5) leads to a nonconvex optimization problem. We follow [19] in which the CCCP procedure [20] was used to find a local optima of a similar objective. CCCP is an iterative algorithm that uses a decomposition of the objective into a sum of convex and concave parts $E(\mathbf{w}) = E_{\text{convex}}(\mathbf{w}) + E_{\text{concave}}(\mathbf{w})$.

$$E_{\text{convex}}(\mathbf{w}) = \frac{1}{2}||\mathbf{w}||^2 + C \sum_{i=1}^{n} \max_{s \in \mathcal{S}(x_i)} [\mathbf{w} \cdot \Phi(x_i, s) + L_{\text{margin}}(y_i, s)] \tag{8}$$

$$E_{\text{concave}}(\mathbf{w}) = -C \sum_{i=1}^{n} \max_{s \in \mathcal{S}(x_i)} [\mathbf{w} \cdot \Phi(x_i, s) - L_{\text{output}}(y_i, s)] \tag{9}$$

In each iteration, CCCP computes a linear upper bound to $E_{\text{concave}}$ based on a current weight vector $\mathbf{w}_t$. The bound depends on subgradients of the summands in (9). For each summand, we take the subgradient $\Phi(x_i, s_i(\mathbf{w}_t))$, where $s_i(\mathbf{w}) = \text{argmax}_{s \in \mathcal{S}(x_i)} [\mathbf{w} \cdot \Phi(x_i, s) - L_{\text{output}}(y_i, s)]$ is a loss augmented prediction.

We note that computing $s_i(\mathbf{w}_t)$ for each training example can be costly. But from our definition of $L_{\text{output}}$, we have that $s_i(\mathbf{w}) = \perp$ for a background example independent of $\mathbf{w}$. Therefore, for a background example $\Phi(x_i, s_i(\mathbf{w}_t)) = 0$.

Table 1: PASCAL 2010 results. UoC-TTI and our method compete in `comp3`. Poselets competes `comp4` due to its use of detailed pose and visibility annotations and non-PASCAL images.

| | Grammar | +bbox | +context | UoC-TTI [9] | +bbox | +context | Poselets [9] |
|---|---|---|---|---|---|---|---|
| **AP** | 47.5 | 47.6 | **49.5** | 44.4 | 45.2 | 47.5 | 48.5 |

Table 2: Training objective and model structure evaluation on PASCAL 2007.

| | Grammar LSVM | Grammar WL-SSVM | Mixture LSVM | Mixture WL-SSVM |
|---|---|---|---|---|
| **AP** | 45.3 | **46.7** | 42.6 | 43.2 |

After computing $s_i(\mathbf{w}_t)$ and $\Phi(x_i, s_i(\mathbf{w}_t))$ for all examples (implicitly for background examples), the weight vector is updated by minimizing a convex upper bound on the objective $E(\mathbf{w})$:

$$\mathbf{w}_{t+1} =$$

$$\operatorname*{argmin}_{\mathbf{w}} \frac{1}{2}||\mathbf{w}||^2 + C \sum_{i=1}^{n} \left[ \max_{s \in \mathcal{S}(x_i)} [\mathbf{w} \cdot \Phi(x_i, s) + L_{\mathrm{margin}}(y_i, s)] - \mathbf{w} \cdot \Phi(x_i, s_i(\mathbf{w}_t)) \right]. \quad (10)$$

The optimization subproblem defined by equation (10) is similar in form to a structural SVM optimization. Given the size and nature of our training dataset we opt to solve this subproblem using stochastic subgradient descent and a modified form of the data mining procedure from [10]. As in [10], we data mine over background images to collect support vectors for background examples. However, unlike in the binary LSVM setting considered in [10], we also need to apply data mining to foreground examples. This would be slow because it requires performing relatively expensive inference (more than 1 second per image) on thousands of images. Instead of applying data mining to the foreground examples, each time we compute $s_i(\mathbf{w}_t)$ for a foreground example, we also compute the top $M$ scoring outputs $s \in \mathcal{S}(x_i)$ of $\mathbf{w}_t \cdot \Phi(x_i, s) + L_{\mathrm{margin}}(y_i, s)$, and place the corresponding feature vectors in the data mining cache. This is efficient since much of the required computation is shared with computation already necessary for computing $s_i(\mathbf{w}_t)$. While this is only a heuristic approximation to true data mining, it leads to an improvement over training with binary LSVM (see Section 5). In practice, we find that $M = 1$ is sufficient for improved performance and that increasing $M$ beyond 1 does not improve our results.

### 4.3 Initialization

Using CCCP requires an initial model or heuristic for selecting the initial outputs $s_i(\mathbf{w}_0)$. Inspired by the methods in [10, 12], we train a single filter for fully visible people using a standard binary SVM. To define the SVM's training data, we select vertically elongated examples. We apply the orientation clustering method in [12] to further divide these examples into two sets that approximately correspond to left-facing versus right-facing orientations. Examples from one of these two sets are then anisotropically rescaled so their HOG feature maps match the dimensions of the filter. These form the positive examples. For negative examples, random patches are extracted from background images. After training the initial filter, we slice it into subfilters (one $8 \times 8$ and five $3 \times 8$) that form the building blocks of the grammar model. We mirror these six filters to get subtypes, and then add subparts using the energy covering heuristic in [10, 12].

## 5 Experimental results

We evaluated the performance of our person grammar and training framework on the PASCAL VOC 2007 and 2010 datasets [8, 9]. We used the standard PASCAL VOC `comp3` test protocol, which measures detection performance by average precision (AP) over different recall levels. Figure 2 shows some qualitative results, including failure cases.

**PASCAL VOC 2010** Our results on the 2010 dataset are presented in Table 1 in the context of two strong baselines. The first, UoC-TTI, won the person category in the `comp3` track of the 2010 competition [9]. The 2010 entry of the UoC-TTI method extended [12] by adding an extra octave to the HOG feature map pyramid, which allows the detector to find smaller objects. We report the AP score of the UoC-TTI "raw" person detector, as well as the scores after applying the bounding

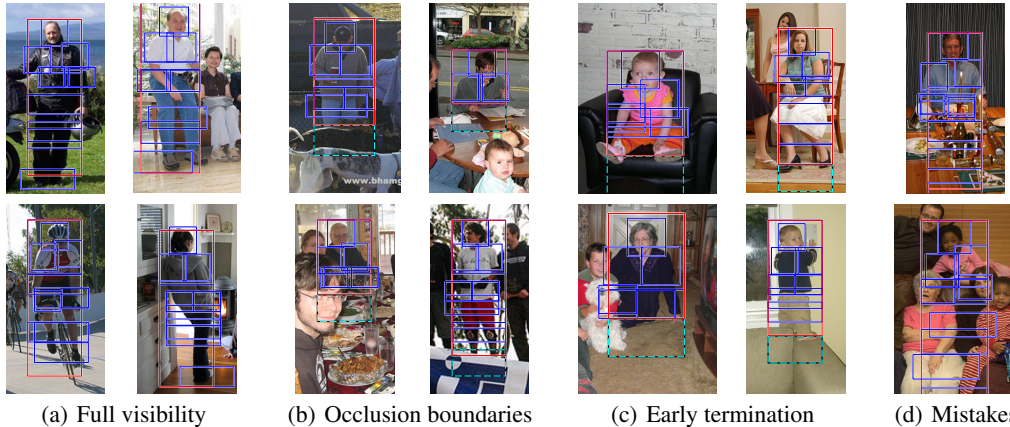

|  (a) Full visibility  |  (b) Occlusion boundaries  |  (c) Early termination  |  (d) Mistakes  |

Figure 2: *Example detections.* Parts are blue. The occlusion part, if used, is dashed cyan. (a) Detections of fully visible people. (b) Examples where the occlusion part detects an occlusion boundary. (c) Detections where there is no occlusion, but a partial person is appropriate. (d) Mistakes where the model did not detect occlusion properly.

box prediction and context rescoring methods described in [10]. Comparing raw detector outputs our grammar model significantly outperforms the mixture model: 47.5 vs. 44.4.

We also applied the two post-processing steps to the grammar model, and found that unlike with the mixture model, the grammar model does not benefit from bounding box prediction. This is likely because our fine-grained occlusion model reduces the number of near misses that are fixed by bounding box prediction. To test context rescoring, we used the UoC-TTI detection data for the other 19 object classes. Context rescoring boosts our final score to 49.5.

The second baseline is the poselets system described in [3]. Their system requires detailed pose and visibility annotations, in contrast to our grammar model which was trained only with bounding box labels. Prior to context rescoring, our model scores one point lower than the poselets model, and after rescoring it scores one point higher.

**Structure and training**  We evaluated several aspects of our model structure and training objective on the PASCAL VOC 2007 dataset. In all of our experiments we set the regularization constant to $C = 0.006$. In Table 2 we compare the WL-SSVM framework developed here with the binary LSVM framework from [10]. WL-SSVM improves performance of the grammar model by 1.4 AP points over binary LSVM training. WL-SSVM also improves results obtained using a mixture of part-based models by 0.6 points. To investigate model structure, we evaluated the effect of part subtypes and occlusion modeling. Removing subtypes reduces the score of the grammar model from 46.7 to 45.5. Removing the occlusion part also decreases the score from 46.7 to 45.5. The shallow model (no subparts) achieves a score of 40.6.

## 6   Discussion

Our results establish grammar-based methods as a high-performance approach to object detection by demonstrating their effectiveness on the challenging task of detecting people in the PASCAL VOC datasets. To do this, we carefully designed a flexible grammar model that can detect people under a wide range of partial occlusion, pose, and appearance variability. Automatically learning the structure of grammar models remains a significant challenge for future work. We hope that our empirical success will provide motivation for pursing this goal, and that the structure of our handcrafted grammar will yield insights into the properties that an automatically learned grammar might require. We also develop a structured training framework, weak-label structural SVM, that naturally handles learning a model with strong outputs, such as derivation trees, from data with weak labels, such as bounding boxes. Our training objective is nonconvex and we use a strong loss function to avoid bad local optima. We plan to explore making this loss softer, in an effort to make learning more robust to outliers.

**Acknowledgments**  This research has been supported by NSF grant IIS-0746569.

## Footnotes

[1] [2] claims the ranking framework overcomes a loss in performance when the number of background examples is increased. In contrast, we don't use a ranking framework but always observed a performance improvement when increasing the number of background examples.

# References

[1] M. Blaschko and C. Lampert. Learning to localize objects with structured output regression. In *ECCV*, 2008.

[2] M. Blaschko, A. Vedaldi, and A. Zisserman. Simultaneous object detection and ranking with weak supervision. In *NIPS*, 2010.

[3] L. Bourdev, S. Maji, T. Brox, and J. Malik. Detecting people using mutually consistent poselet activations. In *ECCV*, 2010.

[4] R. Collobert, F. Sinz, J. Weston, and L. Bottou. Trading convexity for scalability. In *ICML*, 2006.

[5] N. Dalal and B. Triggs. Histograms of oriented gradients for human detection. In *CVPR*, 2005.

[6] C. Do, Q. Le, C. Teo, O. Chapelle, and A. Smola. Tighter bounds for structured estimation. In *NIPS*, 2008.

[7] M. Enzweiler, A. Eigenstetter, B. Schiele, and D. M. Gavrila. Multi-cue pedestrian classification with partial occlusion handling. In *CVPR*, 2010.

[8] M. Everingham, L. Van Gool, C. K. I. Williams, J. Winn, and A. Zisserman. The PASCAL Visual Object Classes Challenge 2007 (VOC2007) Results. http://www.pascal-network.org/challenges/VOC/voc2007/workshop/index.html.

[9] M. Everingham, L. Van Gool, C. K. I. Williams, J. Winn, and A. Zisserman. The PASCAL Visual Object Classes Challenge 2010 (VOC2010) Results. http://www.pascal-network.org/challenges/VOC/voc2010/workshop/index.html.

[10] P. Felzenszwalb, R. Girshick, D. McAllester, and D. Ramanan. Object detection with discriminatively trained part based models. *PAMI*, 2009.

[11] P. Felzenszwalb and D. McAllester. Object detection grammars. *Univerity of Chicago, CS Dept., Tech. Rep. 2010-02*.

[12] P. F. Felzenszwalb, R. B. Girshick, and D. McAllester. Discriminatively trained deformable part models, release 4. http://people.cs.uchicago.edu/˜pff/latent-release4/.

[13] Y. Jin and S. Geman. Context and hierarchy in a probabilistic image model. In *CVPR*, 2006.

[14] D. McAllester and J. Keshet. Generalization bounds and consistency for latent structural probit and ramp loss. In *NIPS*, 2011.

[15] Y. Ohta, T. Kanade, and T. Sakai. An analysis system for scenes containing objects with substructures. In *ICPR*, 1978.

[16] B. Taskar, C. Guestrin, and D. Koller. Max-margin markov networks. In *NIPS*, 2003.

[17] I. Tsochantaridis, T. Joachims, T. Hofmann, and Y. Altun. Large margin methods for structured and interdependent output variables. *JMLR*, 2006.

[18] X. Wang, T. Han, and S. Yan. An hog-lbp human detector with partial occlusion handling. In *ICCV*, 2009.

[19] C.-N. J. Yu and T. Joachims. Learning structural svms with latent variables. In *ICML*, 2009.

[20] A. Yuille and A. Rangarajan. The concave-convex procedure. *Neural Computation*, 2003.

[21] L. Zhu, Y. Chen, A. Torralba, W. Freeman, and A. Yuille. Part and appearance sharing: Recursive compositional models for multi-view multi-object detection. In *CVPR*, 2010.

[22] L. Zhu, Y. Chen, and A. Yuille. Unsupervised learning of probabilistic grammar-markov models for object categories. *PAMI*, 2009.

[23] L. Zhu, Y. Chen, A. Yuille, and W. Freeman. Latent hierarchical structural learning for object detection. In *CVPR*, 2010.

[24] S. Zhu and D. Mumford. A stochastic grammar of images. *Foundations and Trends in Computer Graphics and Vision*, 2006.

